# Visual Development Aids the Acquisition of Motion Velocity Sensitivities

**Robert A. Jacobs**
Department of Brain and Cognitive Sciences
University of Rochester
Rochester, NY 14627
*robbie@bcs.rochester.edu*

**Melissa Dominguez**
Department of Computer Science
University of Rochester
Rochester, NY 14627
*melissad@cs.rochester.edu*

## Abstract

We consider the hypothesis that systems learning aspects of visual perception may benefit from the use of suitably designed developmental progressions during training. Four models were trained to estimate motion velocities in sequences of visual images. Three of the models were "developmental models" in the sense that the nature of their input changed during the course of training. They received a relatively impoverished visual input early in training, and the quality of this input improved as training progressed. One model used a coarse-to-multiscale developmental progression (i.e. it received coarse-scale motion features early in training and finer-scale features were added to its input as training progressed), another model used a fine-to-multiscale progression, and the third model used a random progression. The final model was non-developmental in the sense that the nature of its input remained the same throughout the training period. The simulation results show that the coarse-to-multiscale model performed best. Hypotheses are offered to account for this model's superior performance. We conclude that suitably designed developmental sequences can be useful to systems learning to estimate motion velocities. The idea that visual development can aid visual learning is a viable hypothesis in need of further study.

## 1   Introduction

With relatively few exceptions, relationships between development and learning have largely been ignored by the neural computation community. This is surprising because development may be nature's way of biasing biological learning systems so that they achieve better performance. Development may also represent an effective means for engineers to bias machine learning systems. Learning systems are inherently faced with the bias-variance dilemma [1]. Systems with little or no bias tend to interpolate in unpredictable ways and, thus, have highly variable generalization performance. Systems with larger bias, in contrast, tend to show better generalization performance when exposed to those training sets that they can adequately learn. Development may be an effective means of adding suitable bias to a system thereby enhancing the generalization performance of that system.

In previous work, we studied the effects of different types of developmental sequences on the performances of systems trained to estimate the binocular disparities present in pairs of visual images [2]. Systems consisted of three components. The first component was a pair of right-eye and left-eye images. For example, the images may have depicted a light or dark object against a gray background. The second component was a set of binocular energy filters. These filters are widely used to model the binocular sensitivities of simple and complex cells in primary visual cortex of primates [3]. Based on local patches of the right-eye and left-eye images, each filter acted as a disparity feature detector at a coarse, medium, or fine scale depending on whether the filter was tuned to a low, medium, or high spatial frequency, respectively. The third component was an artificial neural network. The outputs of the binocular energy filters were the inputs to this network. The network was trained to estimate the disparity of the object which was defined as the amount that the object was shifted between the right-eye and left-eye images.

A non-developmental system was compared to three developmental systems. The network of the non-developmental system received the outputs of all binocular energy filters throughout the entire training period. The networks of the developmental systems, in contrast, were trained in three stages. The network of the coarse-to-multiscale system received the outputs of binocular energy filters tuned to a low spatial frequency during the first training stage. It received the outputs of filters tuned to low and medium spatial frequencies during the second training stage, and it received the outputs of all filters during the third training stage. The network of the fine-to-multiscale system was trained in an analogous way, though its filters were added in the opposite order. This network received the outputs of filters tuned to a high frequency during the first training stage, and the outputs of medium and then low frequency filters were added during subsequent stages. The network of the random developmental model was also trained in stages, though its inputs were chosen at random at each stage and, thus, were not organized by spatial frequency content.

The results show that the coarse-to-multiscale and fine-to-multiscale systems consistently outperformed the non-developmental and random developmental systems. The fact that they outperformed the non-developmental system is important because this demonstrates that models that undergo a developmental maturation can acquire a more advanced perceptual ability than one that does not. The fact that they outperformed the random developmental system is important because this demonstrates that not all developmental sequences can be expected to provide performance benefits. To the contrary, only sequences whose characteristics are matched to the task should lead to superior performance. In conjunction with other results not described here, these findings suggest that the most successful systems at learning to detect binocular disparities are systems that are exposed to visual inputs at a single scale early in training, and for which the resolution of their inputs progresses in an orderly fashion from one scale to a neighboring scale during the course of training.

At a more general level, these results suggest that the idea that visual development aids visual learning is a viable hypothesis in need of further study. This paper studies this hypothesis in the context of visual motion velocity estimation. Our simulations show that the tasks of disparity estimation and velocity estimation yield similar, though not identical, patterns of results. Although a developmental approach to the velocity estimation task is shown to be beneficial, it is not the case that all developmental progressions that lead to performance advantages on the disparity estimation task also lead to advantages on the velocity estimation task. In particular, a coarse-to-multiscale developmental system outperformed non-developmental and random developmental systems on the velocity estimation task, but a fine-to-multiscale system did not. We hypothesize that the performance advantage of the coarse-to-multiscale system relative to the fine-to-multiscale system is due to the fact that the coarse-to-multiscale system learned to make greater use of motion energy filters tuned to a low spatial frequency. Analyses suggest that coarse-scale motion features are more informative for the velocity estimation task than fine-scale features.

## 2 Developmental and Non-developmental Systems

The structure of the developmental and non-developmental systems was as follows. The input to each system was a sequence of 88 retinal images where each image was a one-dimensional array 40 pixels in length. As described below, this sequence depicted an object moving at a constant velocity in front of a stationary background. The retinal array was treated as if it were shaped like a circle in the sense that the leftmost and rightmost pixels were regarded as neighbors. This wraparound of the left and right edges was done to

avoid edge artifacts in the spatial dimension. Although a one-dimensional retina is a simplification, its use is justified by the need to keep the simulation times within reason. The sequence of retinal images was filtered using motion energy filters.

Based on neurophysiological results, Adelson and Bergen [4] proposed motion energy filters as a way of modeling the motion sensitivities of simple and complex cells in primary visual cortex. A sequence of one-dimensional images can be represented using a two-dimensional array where one dimension encodes space and the other dimension encodes time. In this case, motion energy filters are two-dimensional filters which extract motion information in local patches of the spatiotemporal space.

The receptive field profile of a simple cell can be described mathematically as a Gabor function which is a sinusoid multiplied by a Gaussian envelope. A quadrature pair of such functions with even and odd phases tuned to leftward (-) and rightward (+) directions of motion is given by

$$g_e^\pm \quad = \quad \frac{1}{2\pi\sigma_x\sigma_t}\exp\{-\frac{x^2}{2\sigma_x^2}-\frac{t^2}{2\sigma_t^2}\}\cos(\omega_x x \pm w_t t) \tag{1}$$

$$g_o^\pm \quad = \quad \frac{1}{2\pi\sigma_x\sigma_t}\exp\{-\frac{x^2}{2\sigma_x^2}-\frac{t^2}{2\sigma_t^2}\}\sin(\omega_x x \pm w_t t) \tag{2}$$

where $x$ and $t$ are the spatial and temporal distances to the center of the Gaussian, $\sigma_x^2$ and $\sigma_t^2$ are the spatial and temporal variances of the Gaussian, and $\omega_x$ and $\omega_t$ are the spatial and temporal frequencies of the sinusoids. The ratio $\omega_t/\omega_x$ determines the orientation of a Gabor function in the spatiotemporal space which, in turn, determines the velocity sensitivity of the function.

The activity of a simple cell is given by the square of the convolution of the cell's receptive field profile with the spatiotemporal pattern. The activities of simple cells with even and odd phases are summed in order to form the activity of a complex cell. This activity is known as a motion energy.

In our simulations, we used a subset of the possible receptive-field locations in the two-dimensional (40 pixels × 88 time frames) spatiotemporal space. This subset formed a 20 × 4 uniform grid such that receptive fields were centered on odd-numbered pixels and odd-numbered time frames. This grid was located in the center of the space with respect to the temporal dimension. An advantage of this choice of locations was that edge artifacts were avoided because all receptive-fields fell entirely within the spatiotemporal space.

Fifteen complex cells corresponding to three spatial frequencies and five temporal frequencies were centered at each receptive-field location. The spatial and temporal frequencies were each separated by an octave. Temporal frequencies were chosen so that the set of cells at each spatial frequency had the same pattern of velocity tunings. Specifically, the sets tuned to low (0.0625 cycles/pixel), medium (0.125 cycles/pixel), and high (0.25 cycles/pixel) spatial frequencies had velocity tunings of 0.25, 0.5, 1.0, 2.0, and 4.0 pixels per time frame. All cells were tuned to rightward motion because we restricted our data sets to only include objects that were moving to the right. A cell's spatial and temporal standard

deviations were set to be inversely proportional to its spatial and temporal frequencies, respectively. The outputs of the complex cells within each spatial frequency band were normalized using a softmax nonlinearity. Consequently, complex cells tended to respond to relative contrast in the image sequence rather than absolute contrast [5] [6].

The normalized outputs of the complex cells were the inputs to an artificial neural network. The network had 1200 input units (the complex cells had 80 receptive-field locations and there were 15 cells at each location). The network's hidden layer contained 18 hidden units which were organized into 3 groups of 6 units each. The connectivity of the hidden units was set so that each group had a limited receptive field, and so that neighboring groups had overlapping receptive fields. A group of hidden units received inputs from thirty-two receptive field locations at the complex cell level, and the receptive fields of neighboring groups overlapped by eight receptive-field locations. The hidden units used a logistic activation function. The output layer consisted of a single linear unit; this unit's output was an estimate of the object velocity depicted in the sequence of retinal images.

The weights of an artificial neural network were initialized to small random values, and were adjusted during the course of training to minimize a sum of squared error cost function using a conjugate gradient optimization procedure [7]. Weight sharing was implemented at the hidden unit level so that corresponding units within each group of hidden units had the same incoming and outgoing weight values, and so that a hidden unit had the same set of weight values from each receptive field location at the complex unit level. This provided the network with a degree of translation invariance, and also dramatically decreased the number of modifiable weight values in the network. It therefore decreased the number of data items needed to train the network, and the amount of time needed to train the network.

Models were trained and tested using separate sets of training and test data items. Each set contained 250 randomly generated items. Training was terminated after 100 iterations through the training set. The results reported below are based on the data items from the test set.

Three developmental systems and one non-developmental system were simulated. The *coarse-to-multiscale* system, or model C2M, was trained using a coarse-to-multiscale developmental sequence which was implemented as follows. The training period was divided into three stages. During the first stage, the neural network portion of the model only received the outputs of complex cells tuned to the low spatial frequency (the outputs of other complex cells were set to zero). During the second stage, the network received the outputs of complex cells tuned to low and medium spatial frequencies; it received the outputs of all complex cells during the third stage. The training of the *fine-to-multiscale* system, or model F2M, was identical to that of model C2M except that its training used a fine-to-multiscale developmental sequence. During the first stage of training, its network received the outputs of complex cells tuned to the high spatial frequency. This network received the outputs of complex cells tuned to high and medium spatial frequencies during the second stage, and received the outputs of all complex cells during the third stage. The training of the *random developmental* system, or model RD, also used a developmental sequence, though this sequence was generated randomly and, thus, was not based on the spatial frequency tunings of the complex cells. The collection of complex cells was randomly partitioned into three equal-sized subsets with the constraint that each subset included one-third of the cells at each receptive-field location. During the first stage of training, the neural network portion of the model only received the outputs of the complex cells in the first subset. It received the outputs of the cells in the first and second subsets during the second stage of training, and received the outputs of all complex cells during the third stage. In contrast, the training period of the *non-developmental* system, or model ND, was not divided into separate stages; its neural network received the outputs of all complex cells throughout the entire training period.

## Solid object data item

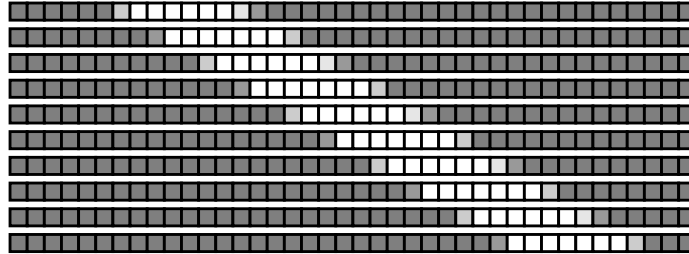

## Noisy object data item

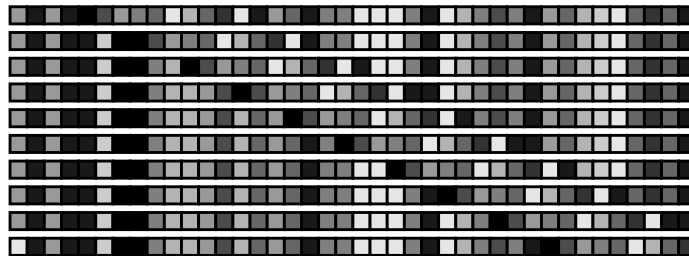

Figure 1: Ten frames of an image sequence from the solid object data set (top) and ten frames of an image sequence from the noisy object data set (bottom).

## 3   Data Sets and Simulation Results

The performances of the four models were evaluated on two data sets. In all cases the images were gray scale with luminance values between 0 and 1, and motion velocities were rightward with magnitudes between 0 and 4 pixels per time frame. Fifteen simulations of each model on each data set were conducted.

In the *solid object data set*, images consisted of a moving light or dark object in front of a stationary gray background. The object's gray-scale values were randomly chosen to either be in the range from 0.0 to 0.1 or from 0.9 and 1.0, whereas the gray-scale value of the background was always 0.5. The size of the object was randomly chosen to be an integer between 6 and 12 pixels, its initial location was a randomly chosen pixel on the retina, and its velocity was randomly chosen to be a real value between 0 and 4 pixels per time frame. Given a sequence of images, the task of a model was to estimate the object's velocity. The top portion of Figure 1 gives an example of ten frames of an image sequence from the solid object data set.

The bar graph in Figure 2 illustrates the results. The horizontal axis gives the model, and the vertical axis gives the root mean squared error (RMSE) on the data items from the test set at the end of training (the error bars give the standard error of the mean). The labels for the developmental models C2M, F2M, and RD include a number. Recall that the training of these models was divided into three training stages (or developmental stages). The number in the label gives the length of developmental stages 1 and 2 (the length of developmental stage 3 can be calculated using the fact that the entire training period lasted 100 iterations). For example, the label 'C2M-5' corresponds to a version of model C2M in which the

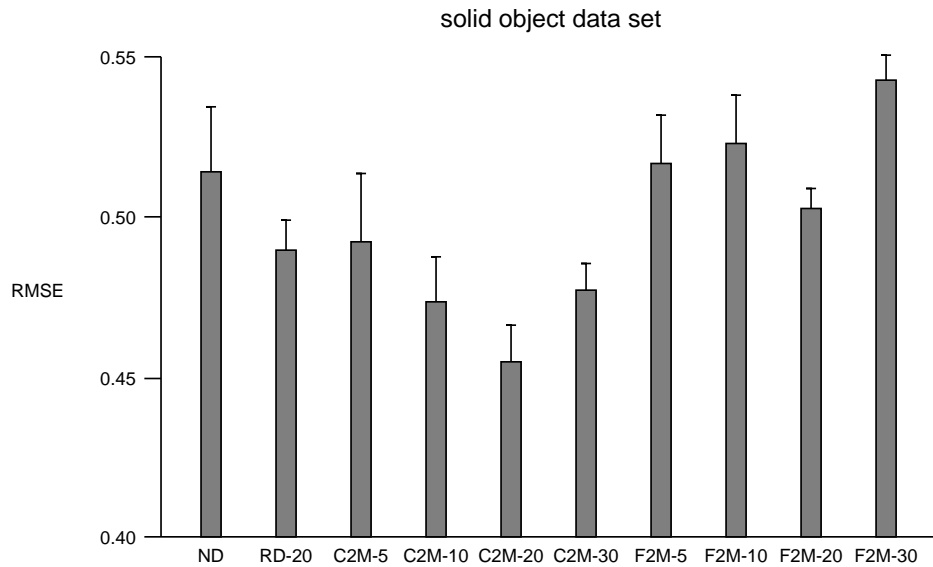

Figure 2: The root mean squared errors (RMSE) on the test set data items for model ND, the best performing version of model RD, and different versions of models C2M and F2M after training on the solid object data set (the error bars give the standard error of the mean).

first stage was 5 iterations, the second stage was 5 iterations, and the third stage was 90 iterations. In regard to model RD, we simulated four versions of this model (RD-5, RD-10, RD-20, and RD-30). For the sake of brevity, only the version that performed best is included in the graph.

Model C2M significantly outperformed all other models. The version of this model which performed best was version C2M-20 which had an 11.5% smaller generalization error than model ND ($t = 2.50$, $p < 0.02$). In addition, C2M-20 had a 9.6% smaller error than the best version of model F2M ($t = 3.57$, $p < 0.01$), and a 7.2% smaller error than the best version of model RD ($t = 2.30$, $p < 0.05$).

The images in the second data set, referred to as the *noisy object data set*, were meant to resemble random-dot kinematograms frequently used in behavioral experiments. Images contained a noisy object which was moving to the right and a noisy background which was stationary. The gray-scale values of the object pixels and the background pixels were set to random numbers between 0 and 1. The size of the object was randomly chosen to be an integer between 6 and 12 pixels, its initial location was a randomly chosen pixel on the retina, and its velocity was randomly chosen to be an integer between 0 and 4 pixels per time frame. As before, the task was to map an image sequence to an estimate of an object velocity. The bottom portion of Figure 1 gives an example of ten frames of an image sequence from the noisy object data set.

The results are shown in Figure 3. Model C2M, once again, outperformed the other models. Relative to model ND, all versions of model C2M showed superior performance (ND vs. C2M-5: $t = 2.69$, $p < 0.02$; ND vs. C2M-10: $t = 2.78$, $p < 0.01$; ND vs. C2M-20: $t = 3.03$, $p < 0.01$; ND vs. C2M-30: $t = 4.14$, $p < 0.001$). The version of model C2M which performed best was version C2M-30. On average, this version had an 8.9% smaller generalization error than model ND, a 6.1% smaller error than the best version of model F2M, and a 4.3% smaller error than the best version of model RD.

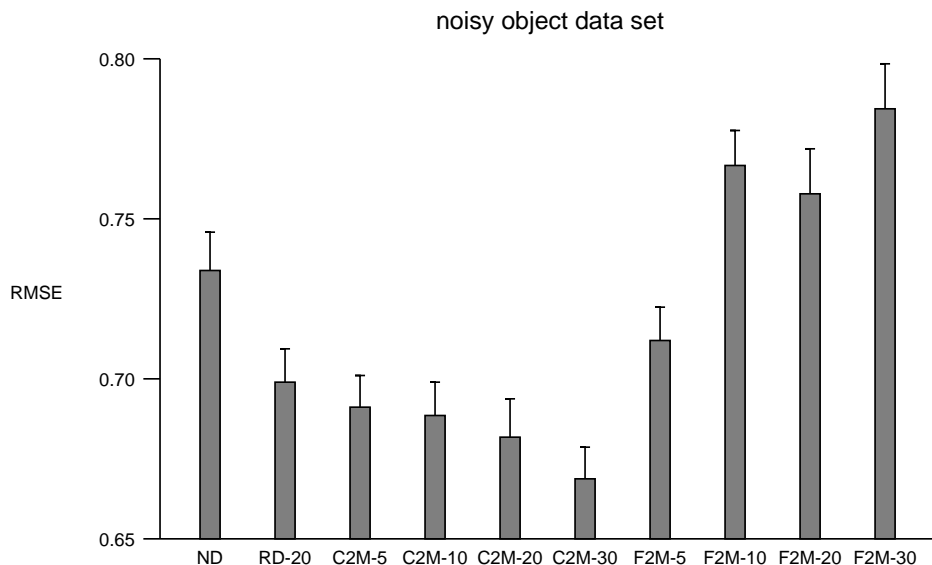

Figure 3: The root mean squared errors (RMSE) on the test set data items for model ND, the best performing version of model RD, and different versions of models C2M and F2M after training on the noisy object data set (the error bars give the standard error of the mean).

Why did model C2M show the best performance? Simulation results described in Jacobs and Dominguez [8] suggest that coarse-scale motion features are more informative for the velocity estimation task than fine-scale features. For example, networks that received only the outputs of complex cells tuned to a low spatial frequency consistently outperformed networks that received only the outputs of mid frequency complex cells or only the outputs of high frequency complex cells. We speculate that coarse-scale motion features are more informative for a number of reasons. First, complex cells tuned to the lowest spatial frequency have the largest receptive fields. As discussed by Weiss and Adelson [9], motion signals tend to be less ambiguous when the stimulus is viewed for a long duration and more ambiguous when the stimulus is viewed for a short duration. This type of reasoning also applies to the activities of complex cells with receptive fields in the spatiotemporal domain. That is, there is comparatively less ambiguity in the activities of complex cells with larger receptive fields than in the activities of cells with smaller receptive fields. Because cells tuned to a low spatial frequency tend to have larger receptive fields than cells tuned to a high spatial frequency, low frequency tuned cells tend to be more reliable for the purposes of motion velocity estimation. Second, model C2M may have benefited from the fact that complex cells with large, overlapping receptive fields provide a high resolution coarse-code of the spatiotemporal space [10]-[12]. This code could provide model C2M with accurate information as to the location of the moving object at each moment in time. For example, the activities of the population of these cells may have coded with high accuracy the fact that the moving object was at location $A$ at time $t_A$ and at location $B$ at time $t_B$. If so, the model's neural network could have easily learned to accurately estimate the object velocity by calculating $(B - A)/(t_B - t_A)$. Model C2M would have an advantage over other models because it received this high resolution coarse-code throughout training. In contrast, model F2M, for example, received early in training only the outputs of complex cells with smaller, less-overlapping receptive fields. The activities of a population of these cells form a lower resolution coarse-code of the spatiotemporal space.

As described above, in earlier work we found that the most successful systems at learning a binocular disparity estimation task were those that: (1) received inputs at a single frequency scale early in training, and (2) for which the resolution of their inputs progressed in an orderly fashion from one scale to a neighboring scale during the course of training [2]. Condition (1) allowed a system to combine and compare input features at an early training stage without the need to compensate for the fact that these features could be at different spatial scales. If condition (2) was satisfied, when a system received inputs at a new spatial scale, it was close to a scale with which the system was already familiar. Although not described here (see Jacobs and Dominguez [8]), we tested the importance on the motion velocity estimation task for the resolution of a system's inputs to progress in an orderly fashion from one scale to a neighboring scale. The results suggest that this factor is moderately important, but not highly important, for a developmental system learning to estimate motion velocities. Overall, it is more important for a system to receive the outputs of the low spatial frequency complex cells as early in training as possible.

Based on the entire set of simulations, we conclude that suitably designed developmental sequences can be useful to systems learning to estimate motion velocities. The idea that visual development can aid visual learning is a viable hypothesis in need of further study.

### Acknowledgments

This work was supported by NIH research grant RO1-EY13149.

## References

[1] Geman, S., Bienenstock, E., and Doursat, R. (1995) Neural networks and the bias/variance dilemma. *Neural Computation*, 4, 1-58.

[2] Dominguez, M. and Jacobs, R.A. (2003) Developmental constraints aid the acquisition of binocular disparity sensitivities. *Neural Computation*, in press.

[3] Ohzawa, I., DeAngelis, G.C., and Freeman, R.D. (1990) Stereoscopic depth discrimination in the visual cortex: Neurons ideally suited as disparity detectors. *Science*, 249, 1037-1041.

[4] Adelson, E.H. and Bergen, J.R. (1985) Spatiotemporal energy models for the perception of motion. *Journal of the Optical Society of America A*, 2, 284-299.

[5] Heeger, D.J. (1992) Normalization of cell responses in cat striate cortex. *Visual Neuroscience*, 9, 181-197.

[6] Nowlan, S.J. and Sejnowski, T.J. (1994) Filter selection model for motion segmentation and velocity integration. *Journal of the Optical Society of America A*, 11, 3177-3200.

[7] Press, W.H., Teukolsky, S.A., Vetterling, W.T., and Flannery, B.P. (1992) *Numerical Recipes in C: The Art of Scientific Computing*. Cambridge, UK: Cambridge University Press.

[8] Jacobs, R.A. and Dominguez, M. (2003) Visual development and the acquisition of motion velocity sensitivities. *Neural Computation*, in press.

[9] Weiss, Y. and Adelson, E.H. (1998) Slow and smooth: A Bayesian theory for the combination of local motion signals in human vision. Center for Biological and Computational Learning Paper Number 158, Massachusetts Institute of Technology, Cambridge, MA.

[10] Milner, P.M. (1974) A model for visual shape recognition. *Psychological Review*, 81, 521-535.

[11] Hinton, G.E. (1981) Shape representation in parallel systems. In A. Drina (Ed.), *Proceedings of the Seventh International Joint Conference on Artificial Intelligence*.

[12] Ballard, D.H. (1986) Cortical connections and parallel processing: Structure and function. *Behavioral and Brain Sciences*, 9, 67-120.
